# Neural network models of chemotaxis in the nematode *Caenorhabditis elegans*

**Thomas C. Ferrée, Ben A. Marcotte, Shawn R. Lockery**
Institute of Neuroscience, University of Oregon, Eugene, Oregon 97403

## Abstract

We train recurrent networks to control chemotaxis in a computer model of the nematode *C. elegans*. The model presented is based closely on the body mechanics, behavioral analyses, neuroanatomy and neurophysiology of *C. elegans*, each imposing constraints relevant for information processing. Simulated worms moving autonomously in simulated chemical environments display a variety of chemotaxis strategies similar to those of biological worms.

## 1 INTRODUCTION

The nematode *C. elegans* provides a unique opportunity to study the neuronal basis of neural computation in an animal capable of complex goal-oriented behaviors. The adult hermaphrodite is only 1 mm long, and has exactly 302 neurons and 95 muscle cells. The morphology of every cell and the location of most electrical and chemical synapses are known precisely (White *et al.*, 1986), making *C. elegans* especially attractive for study. Whole-cell recordings are now being made on identified neurons in the nerve ring of *C. elegans* to determine electrophysiological properties which underly information processing in this animal (Lockery and Goodman, unpublished). However, the strengths and polarities of synaptic connections are not known, so we use neural network optimization to find sets of synaptic strengths which reproduce actual nematode behavior in a simulated worm.

We focus on chemotaxis, the ability to move up (or down) a gradient of chemical attractants (or repellants). In the laboratory, flat Petri dishes (radius = 4.25 cm) are prepared with a Gaussian-shaped field of attractant at the center, and worms are allowed to move freely about. Worms propel themselves forward by generating an undulatory body wave, which produces sinusoidal movement. In chemotaxis, the nervous system generates motor commands which bias this movement and direct

the animal toward higher attractant concentration.

Anatomical constraints pose important problems for *C. elegans* during chemotaxis. In particular, the animal detects the presence of chemicals with a pair of sensory organs (amphids) at the tip of the nose, each containing the processes of multiple chemosensory neurons. During normal locomotion, however, the animal moves on its side so that the two amphids are perpendicular to the Petri dish. *C. elegans* cannot, therefore, sense the gradient directly. One possible strategy for chemotaxis, which has been suggested previously (Ward, 1973), is that the animal computes a temporal derivative of the local concentration during a single head sweep, and combines this with some form of proprioceptive feedback indicating muscle contraction and the direction of head sweep, to compute the spatial gradient for chemotaxis. The existence of this and other strategies is discussed later.

In Section 2, we derive a simple model of the nematode body which produces realistic sinusoidal trajectories in response to motor commands from the nervous system. In Section 3, we give a simple model of the *C. elegans* nervous system based on preliminary physiological data. In Section 4, we use a stochastic optimization algorithm to determine sets of synaptic weights which control chemotaxis, and discuss solutions.

## 2   BIOMECHANICS OF NEMATODE ORIENTATION

Nematode locomotion has been studied in detail (Niebur and Erdös, 1991; Niebur and Erdös, 1993). These authors derived Newtonian force equations for each muscular segment of the body, which can be solved numerically to generate forward sinusoidal movement. Unfortunately, such a thorough treatment is computationally intensive and not practical to use with network optimization. To simplify the problem we first recognize that chemotaxis is a behavior more of orientation than of locomotion. We therefore derive a set of biomechanical equations which direct the head to generate sinusoidal movement, which can be biased by the network toward higher chemical concentrations.

We focus our attention on the point $(x, y)$ at the tip of the nose, since that is where the animal senses the chemical environment. As shown in Figure 1(a), we assign a velocity vector $\vec{v}$ directed along the midline of the first body segment, *i.e.*, the head. Assuming that the worm moves forward at constant speed $v$, we can write the velocity vector as

$$\vec{v}(t) \equiv \left( \frac{dx}{dt}, \frac{dy}{dt} \right) = \Big( v \cos \theta(t), v \sin \theta(t) \Big) \tag{1}$$

where $x$, $y$ and $\theta$ are measured relative to fixed coordinates in the Petri dish. Assuming that the worm moves without lateral slipping and that the undulatory wave of muscular contraction initiated in the neck travels posteriorly without modification, then each body segment simply follows the one previous (anterior) to it. In this way, the head directs the movement and the rest of the body simply follows.

Figure 1(b) shows an expanded view of the neck segment. As the worm moves forward, the posterior boundary of that segment assumes the position held by its anterior neighbor at a slightly earlier time. If $L$ is the total body length and $N$ is

the number of body segments, then this time delay is $\delta t \simeq L/Nv$. (For $L = 1$ mm, $v = 0.22$ mm/s and $N = 10$ we have $\delta t \simeq 0.45$ s, roughly an order of magnitude smaller than the relevant behavioral time scale: the head-sweep period $T \simeq 4.2$ s.) If we define the neck angle $\alpha(t) \equiv \theta_1(t) - \theta_2(t)$, then the above arguments imply

$$\alpha(t) = \theta_1(t) - \theta_1(t - \delta t) \simeq \frac{d\theta_1}{dt}\, \delta t \tag{2}$$

where the second relation is essentially a backward-Euler algorithm for $d\theta_1/dt$. Since $\theta \equiv \theta_1$, we have reached the intuitive result that the neck angle $\alpha$ determines the rate of turning $d\theta/dt$. Note that while $\theta_1$ and $\theta_2$ are defined relative to the fixed laboratory coordinates, their difference $\alpha$ is invariant under rotations of these coordinates, and can therefore be viewed as intrinsic to the body. This allows us to derive an expression for $\alpha$ in terms of muscle cell contraction, or motor neuron depolarization, as follows.

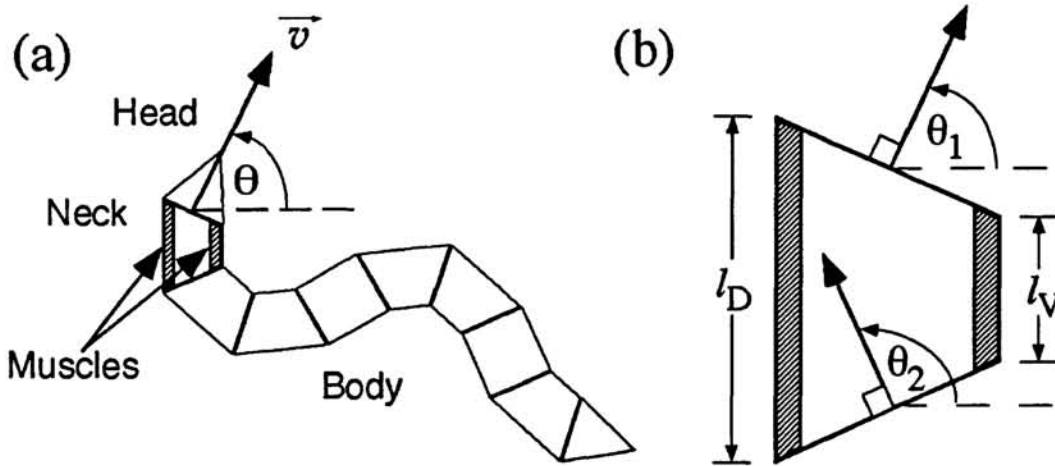

Figure 1: Nematode body mechanics. (a) Segmented model of the nematode body, showing the direction of motion $\vec{v}$. (b) Expanded view of the neck segment, showing dorsal (D) and ventral (V) neck muscles.

Nematodes maintain nearly constant volume during movement. To incorporate this constraint, albeit approximately, we assume that at all times the geometry of each segment is such that $(l_D - l_0) = -(l_V - l_0)$, where $l_0 \equiv L/N$ is the equilibrium length of a relaxed segment. For small angles $\alpha$, we have $\alpha \simeq (l_V - l_D)/d$, where $d$ is the body diameter. The dashed lines in Figure 1(b) indicate dorsal and ventral muscles, which are believed to develop tension nearly independent of length (Toida *et al.*, 1975). When contracting, these muscles must work against the elasticity of the cuticle, internal fluid pressure, and elasticity and developed tension of the opposing muscles. If these elastic forces act linearly, then $T_D - T_V \simeq k\,(l_V - l_D)$, where $T_D$ and $T_V$ are dorsal and ventral muscle tensions, and $k$ is an effective force constant. For simplicity, we further assume that each muscle develops tension linearly in response to the voltage of its corresponding motor neuron, *i.e.*, $T_{D,V} = \epsilon\, V_{D,V}$, where $\epsilon$ is a positive constant, and $V_D$ and $V_V$ are dorsal and ventral motor neuron voltages.

Combining these results, we have finally

$$\frac{d\theta}{dt} = \gamma \left( V_D(t) - V_V(t) \right) \tag{3}$$

where $\gamma = (Nv/L) \cdot (\epsilon/kd)$. With appropriate motor commands, equations (1) and (3) can be integrated numerically to generate sinusoidal worm trajectories like those of biological worms. This model embodies the main anatomical features that are likely to be important in *C. elegans* chemotaxis, yet is sufficiently compact to be embedded in a network optimization procedure.

## 3  CHEMOTAXIS CONTROL CIRCUIT

*C. elegans* neurons are tiny and have very simple morpologies: a typical neuron in the head has a spherical soma 1–2 $\mu$m in diameter, and a single cylindrical process 60–80 $\mu$m in length and 0.1–0.2 $\mu$m in diameter. Compartmental models, based on this morphology and preliminary physiological recordings, indicate that *C. elegans* neurons are effectively isopotential (Lockery, 1995). Furthermore, *C. elegans* neurons do not fire classical all-or-none action potentials, but appear to rely primarily on graded signal propagation (Lockery and Goodman, unpublished). Thus, a reasonable starting point for a network model is to represent each neuron by a single isopotential compartment, in which voltage is the state variable, and the membrane conductance in purely ohmic.

Anatomical data indicate that the *C. elegans* nervous system has both electrical and chemical synapses, but the synaptic transfer functions are not known. However, steady-state synaptic transfer functions for chemical synapses have been measured in *Ascaris suum*, a related species of nematode, where it was found that postsynaptic voltage is a graded function of presynaptic voltage, due to tonic neurotransmitter release (Davis and Stretton, 1989). This voltage dependence is sigmoidal, *i.e.*, $V_{\text{post}} \sim \tanh(V_{\text{pre}})$. A simple network model which captures all of these features is

$$\tau \frac{dV_i}{dt} = -V_i + V_{\text{max}} \tanh\left(\beta \sum_{j=1}^{n} w_{ij} \left(V_j - \bar{V}_j\right)\right) + V_i^{\text{stim}}(t) \qquad (4)$$

where $V_i$ is the voltage of the $i^{\text{th}}$ neuron. Here all voltages are measured relative to a common resting potential, $V_{\text{max}}$ is an arbitrary voltage scale which sets the operational range of the neurons, and $\beta$ sets the voltage sensitivity of the synaptic transfer function. The weight $w_{ij}$ represents the net strength and polarity of all synaptic connections from neuron $j$ to neuron $i$, and the constants $\bar{V}_j$ determine the center of each transfer function. The membane time constant $\tau$ is assumed to be the same for all cells, and will be discussed further later. Note that in (4), synaptic transmission occurs instantaneously: the time constant $\tau$ arises from capacitive current through the cell membrane, and is unrelated to synaptic transmission. Note also that the way in which (4) sums multiple inputs is not unique, *i.e.*, other sigmoidal models which sum inputs differently are equally plausible, since no data on synaptic summation exists for either *C. elegans* or *Ascaris suum.*

The stimulus term $V_i^{\text{stim}}(t)$ is used to introduce chemosensation and sinusoidal locomotion to the network in (4). We use $i = 1$ to label a single chemosensory neuron at the tip of the nose, and $i = n - 1 \equiv D$ and $i = n \equiv V$ to label dorsal and ventral motor neurons. For simplicity we assume that the chemosensory neuron voltage responds linearly to the local chemical concentration:

$$V_1^{\text{stim}}(t) = V_{\text{chem}} \, C(x, y) \qquad (5)$$

where $V_{\text{chem}}$ is a positive constant, and the local concentration $C(x,y)$ is always evaluated at the instantaneous nose position.

In the previous section, we emphasized that locomotion is effectively independent of orientation. We therefore assume the existence of a central pattern generator (CPG) which is *outside* the chemotaxis control circuit (4). Thus, in addition to synaptic input from other neurons, each motor neuron receives a sinusoidal stimulus

$$V_{\text{D}}^{\text{stim}}(t) = -V_{\text{V}}^{\text{stim}}(t) = V_{\text{CPG}} \; \sin(\omega t) \qquad (6)$$

where $V_{\text{CPG}}$ and $\omega \equiv 2\pi/T$ are positive constants.

## 4  RESULTS AND DISCUSSION

Equations (1), (3) and (4), together with (5) and (6), comprise a set of $n + 3$ first-order nonlinear differential equations, which can be solved numerically given initial conditions and a specification of the chemical environment. We use a fourth-order Runge-Kutta algorithm and find favorable stability and convergence. The necessary body parameters have been measured by observing actual worms (Pierce and Lockery, unpublished): $v = 0.022$ cm/s, $T = 4.2$ s and $\gamma = 0.8/(2V_{\text{CPG}})$. The chemical environment is also chosen to agree roughly with experimental values: $C(x,y) = C_0 \exp(-(x^2 + y^2)/\lambda_C^2)$, with $C_0 = 0.052$ $\mu$mol/cm$^3$ and $\lambda_C = 2.3$ cm.

To optimize networks to control chemotaxis, we use a simple simulated annealing algorithm which searches over the $(n^2 + 3)$-dimensional space of parameters $w_{ij}$, $\beta$, $V_{\text{chem}}$ and $V_{\text{CPG}}$. In the results shown here, we used $n = 12$, and set $\bar{V}_j = 0$. Each set of the resulting parameters represents a different nervous system for the model worm. At the beginning of each run, the worm is initialized by choosing an initial position $(x_0, y_0)$, an initial angle $\theta_0$, and by setting $V_i = 0$. Upon numerically integrating, simulated worms move autonomously in their environment for a predetermined amount of time, typically the real-time equivalent of 10-15 minutes. We quantify the performance, or fitness, of each worm during chemotaxis by computing the average chemical concentration at the tip of its nose over the duration of each run. To avoid lucky scores, the actual score for each worm is obtained by averaging over several initial conditions.

In Figure 2, we show a comparison of tracks produced by (a) biological and (b) simulated worms during chemotaxis. In each case, three worms were placed in a dish with a radial gradient and allowed to move freely for the real-time equivalent of 15 minutes. In (b), the three worms have the same neural parameters ($w_{ij}$, $\beta$, $V_{\text{chem}}$, $V_{\text{CPG}}$), but different initial angles $\theta_0$. In both (a) and (b), all three worms make initial movements, then move toward the center of the dish and remain there. In other optimizations, rather than orbit the center, the simulated worms may approach the center asymptotically from one side, make simple geometric patterns which pass through the center, or exhibit a variety of other distinct strategies for chemotaxis. This is similar to the situation with biological worms, which also have considerable variation in the details of their tracks.

The behavior shown in Figure 2 was produced using $\tau = 500$ ms. However, preliminary electrophysiological recordings from *C. elegans* neurons suggest that the actual value may be as much as an order of magnitude smaller, but not bigger (Lockery and Goodman, unpublished). This presents a potential problem for chemotaxis com-

putation, since shorter time constants require greater sensitivity to small changes in $C(x, y)$ in order to compute a temporal derivative, which is believed to be required. During optimization, we have seen that for a fixed number of neurons $n$, finding optimal solutions becomes more difficult as $\tau$ is decreased. This observation is very difficult to quantify, however, due to the existence of local maxima in the fitness function. Nevertheless, this suggests that additional mechanisms may need to be included to understand neural computation in *C. elegans*. First, time- and voltage-dependent conductances will modify the effective membrane time constant, and may increase the effective time scale for computation by individual neurons. Second, more neurons and synaptic delays will also move the effective neuronal time scale closer to that of the behavior. Either of these will allow comparisons of $C(x, y)$ across greater distances, thereby requiring less sensitivity to compute the gradient, and potentially improving the ability of these networks to control chemotaxis.

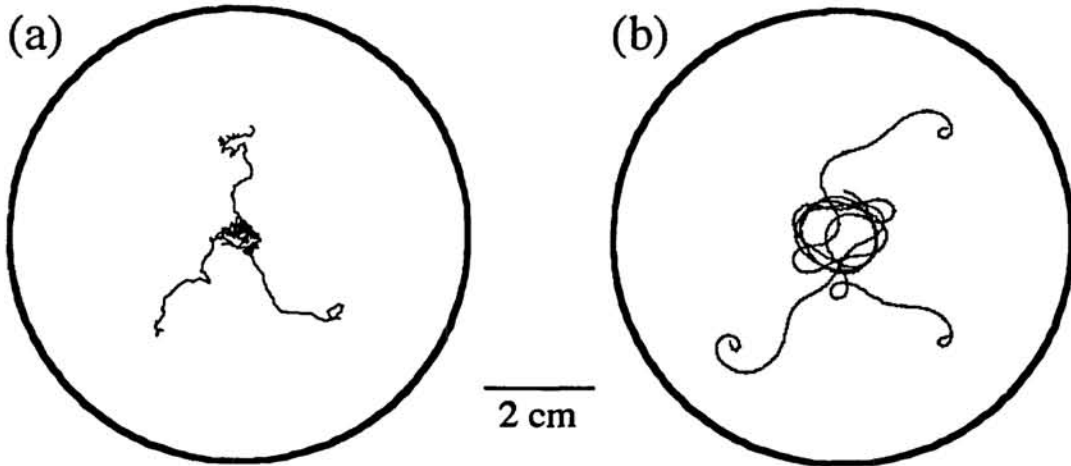

Figure 2: Nematodes performing chemotaxis: (a) biological (Pierce and Lockery, unpublished), and (b) simulated.

We also note, based on a variety of other results, not shown here, that the head-sweep strategy, described in the introduction, is by no means the only strategy for chemotaxis in this system. In particular, we have optimized networks without a CPG, *i.e.*, with $V_{CPG} = 0$ in (6), and found parameter sets that successfully control chemotaxis. This presents the possibility that even worms with a CPG do not necessarily compute the gradient based on lateral movement of the head, but may instead respond only to changes in concentration along their mean trajectory. Similar results have been reported previously, although based on a somewhat different biomechanical model (Beer and Gallagher, 1992).

Finally, we have also optimized discrete-time networks, obtained by setting $\tau = 0$ in (4) and updating all units synchronously. As is well-known, on relatively short time scales ($\sim T$) such a system tends to "overshoot" at each successive time step, leading to sporadic behavior of the network and the body. Knowing this, it is interesting that simulated worms with such a nervous system are capable of reliable behavior over longer time scales, *i.e.*, they successfully perform chemotaxis.

## 5 CONCLUSIONS AND FUTURE WORK

The main result of this paper is that a small nervous system, based on graded-potential neurons, is capable of controlling chemotaxis in a worm-like physical body with the dimensions of *C. elegans*. The model presented is based closely on the body mechanics, behavioral analyses, neuroanatomy and neurophysiology of *C. elegans*, and is a reliable starting point for more realistic models to follow. Furthermore, we have established the existence of chemotaxis strategies that had not been anticipated based on behavioral experiments with real worms.

Future work will involve both improvement of the model and analysis of the resulting solutions. Improvements will include introducing voltage- and time-dependent membrane conductances, as this data becomes available, and more realistic models of synaptic transmission. Also, laser ablation experiments have been performed that suggest which interneurons and motor neurons in *C. elegans* may be important for chemotaxis (Bargmann, unpublished), and these data can be used to constrain the synaptic connections during optimization. Analyses will be aimed at determining the role of individual physiological and anatomical features, and how they function together to govern the collective properties of the network as a whole during chemotaxis.

### Acknowledgements

The authors would like to thank Miriam Goodman and Jon Pierce for helpful discussions. This work has been supported by NIMH MH11373, NIMH MH51383, NSF IBN 9458102, ONR N00014-94-1-0642, the Sloan Foundation, and The Searle Scholars Program.

### References

Beer, R. D. and J. C. Gallagher (1992). Evolving dynamical neural networks for adaptive behavior, *Adaptive Behavior* 1(1):91-122.

Davis, R. E. and A. O. W. Stretton (1989). Signaling properties of *Ascaris* motorneurons: Graded active responses, graded synaptic transmission, and tonic transmitter release, *J. Neurosci.* 9:415-425.

Lockery, S. R. (1995). Signal propagation in the nerve ring of *C. elegans*, *Soc. Neurosci. Abstr.* **569.7**:1454.

Niebur, E. and P. Erdös (1991). Theory of the locomotion of nematodes: Dynamics of undulatory progression on a surface, *Biophys. J.* 60:1132-1146.

Niebur, E. and P. Erdös (1993). Theory of the locomotion of nematodes: Control of the somatic motor neurons by interneurons, *Math. Biosci.* 118:51-82.

Toida, N., H. Kuriyama, N. Tashiro and Y. Ito (1975). Obliquely striated muscle, *Physiol. Rev.* 55:700-756.

Ward, S. (1973). Chemotaxis by the nematode *Caenorhabditis elegans*: Identification of attractants and analysis of the response by use of mutants, *Proc. Nat. Acad. Sci. USA* 70:817-821.

White, J. G., E. Southgate, J. N. Thompson and S. Brenner (1986). The structure of the nervous system of *C. elegans*, *Phil. Trans. R. Soc. London* 314:1-340.